# Semi-supervised protein classification using cluster kernels

**Jason Weston**[*]
Max Planck Institute for Biological Cybernetics,
72076 Tübingen, Germany
weston@tuebingen.mpg.de

**Christina Leslie**
Department of Computer Science,
Columbia University
cleslie@cs.columbia.edu

**Dengyong Zhou, Andre Elisseeff**
Max Planck Institute for Biological Cybernetics,
72076 Tübingen, Germany
zhou@tuebingen.mpg.de

**William Stafford Noble**
Department of Genome Sciences
University of Washington
noble@gs.washington.edu

## Abstract

A key issue in supervised protein classification is the representation of input sequences of amino acids. Recent work using string kernels for protein data has achieved state-of-the-art classification performance. However, such representations are based only on labeled data — examples with known 3D structures, organized into structural classes — while in practice, unlabeled data is far more plentiful. In this work, we develop simple and scalable cluster kernel techniques for incorporating unlabeled data into the representation of protein sequences. We show that our methods greatly improve the classification performance of string kernels and outperform standard approaches for using unlabeled data, such as adding close homologs of the positive examples to the training data. We achieve equal or superior performance to previously presented cluster kernel methods while achieving far greater computational efficiency.

## 1   Introduction

A central problem in computational biology is the classification of proteins into functional and structural classes given their amino acid sequences. The 3D structure that a protein assumes after folding largely determines its function in the cell. However, it is far easier to determine experimentally the primary sequence of a protein than it is to solve the 3D structure. Through evolution, structure is more conserved than sequence, so that detecting even very subtle sequence similarities, or remote homology, is important for predicting function.

The major methods for homology detection can be split into three basic groups: pairwise sequence comparison algorithms [1, 2], generative models for protein families [3, 4], and discriminative classifiers [5, 6, 7]. Popular sequence comparison methods such as BLAST

---

[*]Supplemental information for the paper, including the data sets and Matlab source code can be found on this author's web page at http://www.kyb.tuebingen.mpg.de/bs/people/weston/semiprot

and Smith-Waterman are based on unsupervised alignment scores. Generative models such as profile hidden Markov models (HMMs) model positive examples of a protein family, but they can be trained iteratively using both positively labeled and unlabeled examples by pulling in close homologs and adding them to the positive set. A compromise between these methods is PSI-BLAST [8], which uses BLAST to iteratively build a probabilistic profile of a query sequence and obtain a more sensitive sequence comparison score. Finally, classifiers such as SVMs use both positive and negative examples and provide state-of-the-art performance when used with appropriate kernels [5, 6, 7]. However, these classifiers still require an auxiliary method (such as PSI-BLAST) to handle unlabeled data: one generally adds predicted homologs of the positive training examples to the training set before training the classifier.

In practice, relatively little labeled data is available — approximately 30,000 proteins with known 3D structure, some belonging to families and superfamilies with only a handful of labeled members — whereas there are close to one million sequenced proteins, providing abundant unlabeled data. New semi-supervised learning techniques should be able to make better use of this unlabeled data.

Recent work in semi-supervised learning has focused on changing the representation given to a classifier by taking into account the structure described by the unlabeled data [9, 10, 11]. These works can be viewed as cases of *cluster kernels*, which produce similarity metrics based on the cluster assumption: namely, two points in the same "cluster" or region of high density should have a small distance to each other. In this work, we investigate the use of cluster kernels for protein classification by developing two simple and scalable methods for modifying a base kernel. The *neighborhood kernel* uses averaging over a neighborhood of sequences defined by a local sequence similarity measure, and the *bagged kernel* uses bagged clustering of the full sequence data set to modify the base kernel. In both the semi-supervised and transductive settings, these techniques greatly improve classification performance when used with mismatch string kernels, and the techniques achieve equal or superior results to all previously presented cluster kernel methods that we tried. Moreover, the neighborhood and bagged kernel approaches are far more computationally efficient than these competing methods.

## 2  Representations and kernels for protein sequences

Proteins can be represented as variable length sequences, typically several hundred characters long, from the alphabet of 20 amino acids. In order to use learning algorithms that require vector inputs, we must first find a suitable feature vector representation, mapping sequence $x$ into a vector space by $x \mapsto \Phi(x)$. If we use kernel methods such as SVMs, which only need to compute inner products $K(x, y) = \langle \Phi(x), \Phi(y) \rangle$ for training and testing, then we can accomplish the above mapping using a kernel for sequence data.

Biologically motivated sequence comparison scores, like Smith-Waterman or BLAST, provide an appealing representation of sequence data. The Smith-Waterman (SW) algorithm [2] uses dynamic programming to compute the optimal local gapped alignment score between two sequences, while BLAST [1] approximates SW by computing a heuristic alignment score. Both methods return empirically estimated E-values indicating the confidence of the score. These alignment-based scores do not define a positive definite kernel; however, one can use a feature representation based on the empirical kernel map

$$\Phi(x) = \langle d(x_1, x), \ldots, d(x_m, x) \rangle$$

where $d(x, y)$ is the pairwise score (or E-value) between $x$ and $y$ and $x_i$, $i = 1 \ldots m$, are the training sequences. Using SW E-values in this fashion gives strong classification performance [7]. Note, however, that the method is slow, both because computing each SW score is $O(|x|^2)$ and because computing each empirically mapped kernel value is $O(m)$.

Another appealing idea is to derive the feature representation from a generative model for a protein family. In the Fisher kernel method [5], one first builds a profile HMM for the positive training sequences, defining a log likelihood function $\log P(x|\theta)$ for any protein sequence $x$. Then the gradient vector $\nabla_\theta \log P(x|\theta)|_{\theta=\theta_0}$, where $\theta_0$ is the maximum likelihood estimate for model parameters, defines an explicit vector of features, called Fisher scores, for $x$. This representation gives excellent classification results, but the Fisher scores must be computed by an $O(|x|^2)$ forward-backward algorithm, making the kernel tractable but slow.

It is possible to construct useful kernels directly without explicitly depending on generative models by using string kernels. For example, the mismatch kernel [6] is defined by a histogram-like feature map that uses mismatches to capture inexact string matching. The feature space is indexed by all possible $k$-length subsequences $\alpha = a_1 a_2 \ldots a_k$, where each $a_i$ is a character in the alphabet $\mathcal{A}$ of amino acids. The feature map is defined on $k$-gram $\alpha$ by $\Phi(\alpha) = (\phi_\beta(\alpha))_{\mathcal{A}^k}$ where $\phi_\beta(\alpha) = 1$ if $\alpha$ is within $m$ mismatches of $\beta$, 0 otherwise, and is extended additively to longer sequences: $\Phi(x) = \sum_{k\text{-grams}\in x} \Phi(\alpha)$. The mismatch kernel can be computed efficiently using a trie data structure: the complexity of calculating $K(x,y)$ is $O(c_K(|x|+|y|))$, where $c_K = k^{m+1}|\mathcal{A}|^m$. For typical kernel parameters $k = 5$ and $m = 1$ [6], the mismatch kernel is fast, scalable and yields impressive performance. Many other interesting models and examples of string kernels have recently been presented. A survey of related string kernel work is given in the longer version of this paper.

String kernel methods with SVMs are a powerful approach to protein classification and have consistently performed better than non-discriminative techniques [5, 7, 6]. However, in a real-world setting, protein classifiers have access to unlabeled data. We now discuss how to incorporate such data into the representation given to SVMs via the use of cluster kernels.

## 3 Cluster kernels for protein sequences

In semi-supervised learning, one tries to improve a classifier trained on labeled data by exploiting (a relatively large set of) unlabeled data. An extensive review of techniques can be found in [12]. It has been shown experimentally that under certain conditions, the decision function can be estimated more accurately in a semi-supervised setting, yielding lower generalization error. The most common assumption one makes in this setting is called the "cluster assumption," namely that *the class does not change in regions of high density*.

Although classifiers implement the cluster assumption in various ways, we focus on classifiers that re-represent the given data to reflect structure revealed by unlabeled data. The main idea is to change the distance metric so that the relative distance between two points is smaller if the points are in the same cluster. If one is using kernels, rather than explicit feature vectors, one can modify the kernel representation by constructing a cluster kernel. In [10], a general framework is presented for producing cluster kernels by modifying the eigenspectrum of the kernel matrix. Two of the main methods presented are the *random walk kernel* and the *spectral clustering kernel*.

The random walk kernel is a normalized and symmetrized version of a transition matrix corresponding to a $t$-step random walk. The random representation described in [11] interprets an RBF kernel as a transition matrix of a random walk on a graph with vertices $x_i$, $P(x_i \rightarrow x_j) = \frac{K_{ij}}{\sum K_{ip}}$. After $t$ steps, the probability of going from a point $x_i$ to a point $x_j$ should be high if the points are in the same cluster. This transition probability can be calculated for the entire matrix as $P^t = (D^{-1}K)^t$, where $D$ is a diagonal matrix such that $D_{ii} = \sum_p K_{ip}$. To obtain a kernel, one performs the following steps. Com-

pute $L = D^{-1/2}KD^{-1/2}$ and its eigendecomposition $L = U\Lambda U^\top$. let $\lambda_i \leftarrow \lambda_i^t$, where $\lambda_i = \Lambda_{ii}$, and let $\tilde{L} = U\tilde{\Lambda}U^\top$. Then the new kernel is $\tilde{K} = \tilde{D}^{1/2}\tilde{L}\tilde{D}^{1/2}$, where $\tilde{D}$ is a diagonal matrix with $\tilde{D}_{ii} = 1/L_{ii}$.

The spectral clustering kernel is a simple use of the representation derived from spectral clustering [13] using the first $k$ eigenvectors. One computes the eigenvectors $(v_1, \ldots, v_k)$ of $D^{-\frac{1}{2}}KD^{-\frac{1}{2}}$, with $D$ defined as before, giving the representation $\phi(x_i)_p = v_{pi}$. This vector can also then be normalized to have length 1. This approach has been shown to produce a well-clustered representation. While in spectral clustering, one then performs $k$-means in this representation, here one simply gives the representation as input to a classifier.

A serious problem with these methods is that one must diagonalize a matrix the size of the set of labeled and unlabeled data. Other methods of implementing the cluster assumption such as transductive SVMs [14] also suffer from computational efficiency issues. A second drawback is that these kernels are better suited to a *transductive* setting (where one is given both the unlabeled and test points in advance) rather than a semi-supervising setting. In order to estimate the kernel for a sequence not present during training, one is forced to solve a difficult regression problem [10]. In the next two sections we will describe two simple methods to implement the cluster assumption that do not suffer from these issues.

## 4   The neighborhood mismatch kernel

In most current learning applications for prediction of protein properties, such as prediction of three-state secondary structure, neural nets are trained on probabilistic *profiles* of a sequence window — a matrix of position-specific emission and gap probabilities — learned from a PSI-BLAST alignment rather than an encoding of the sequence itself. In this way, each input sequence is represented probabilistically by its "neighborhood" in a large sequence database, where PSI-BLAST neighbors are sequences that are closely related through evolution. We wish to transfer the notion of profiles to our mismatch representation of protein sequences.

We use a standard sequence similarity measure like BLAST or PSI-BLAST to define a neighborhood $\text{Nbd}(x)$ for each input sequence $x$ as the set of sequences $x'$ with similarity score to $x$ below a fixed E-value threshold, together with $x$ itself. Now given a fixed original feature representation, we represent $x$ by the average of the feature vectors for members of its neighborhood: $\Phi_{nbd}(x) = \dfrac{1}{|\text{Nbd}(x)|} \sum_{x' \in \text{Nbd}(x)} \Phi_{orig}(x')$. The neighborhood kernel is then defined by:

$$K_{nbd}(x, y) = \frac{1}{|\text{Nbd}(x)||\text{Nbd}(y)|} \sum_{x' \in \text{Nbd}(x), y' \in \text{Nbd}(y)} K_{orig}(x', y').$$

We will see in the experimental results that this simple neighborhood-averaging technique, used in a semi-supervised setting with the mismatch kernel, dramatically improves classification performance.

To see how the neighborhood approach fits with the cluster assumption, consider a set of points in feature space that form a "cluster" or dense region of the data set, and consider the region $R$ formed by the union of the convex hulls of the neighborhood point sets. If the dissimilarity measure is a true distance, the neighborhood averaged vector $\Phi_{nbd}(x)$ stays inside the convex hull of the vectors in its neighborhood, all the neighborhood vectors stay within region $R$. In general, the cluster contracts inside $R$ under the averaging operation. Thus, under the new representation, different clusters can become better separated from each other.

## 5 The bagged mismatch kernel

There exist a number of clustering techniques that are much more efficient than the methods mentioned in Section 3. For example, the classical $k$-means algorithm is $O(rkmd)$, where $m$ is the number of data points, $d$ is their dimensionality, and $r$ is the number of iterations required. Empirically, this running time grows sublinearly with $k$, $m$ and $d$. In practice, it is computationally efficient even to run $k$-means multiple times, which can be useful since $k$-means can converge to local minima. We therefore consider the following method:

1. Run $k$-means $n$ times, giving $p = 1, \ldots, n$ cluster assignments $c_p(x_i)$ for each $i$.
2. Build a bagged-clustering representation based upon the fraction of times that $x_i$ and $x_j$ are in the same cluster:

$$K_{bag}(x_i, x_j) = \frac{\sum_p [c_p(x_i) = c_p(x_j)]}{n}. \tag{1}$$

3. Take the product between the original and bagged kernel:

$$K(x_i, x_j) = K_{orig}(x_i, x_j) \cdot K_{bag}(x_i, x_j)$$

Because $k$-means gives different solutions on each run, step (1) will give different results; for other clustering algorithms one could sub-sample the data instead. Step (2) is a valid kernel because it is the inner product in an $nk$-dimensional space $\Phi(x_i) = \langle [c_p(x_i) = q] : p = 1, \ldots, n, q = 1, \ldots, k \rangle$, and products of kernels as in step (3) are also valid kernels. The intuition behind the approach is that the original kernel is rescaled by the "probability" that two points are in the same cluster, hence encoding the cluster assumption. To estimate the kernel on a test sequence $x$ in a semi-supervised setting, one can assign $x$ to the nearest cluster in each of the bagged runs to compute $K_{bag}(x, x_i)$. We apply the bagged kernel method with $K_{orig}$ as the mismatch kernel and $K_{bag}$ built using PSI-BLAST.

## 6 Experiments

We measure the recognition performance of cluster kernels methods by testing their ability to classify protein domains into superfamilies in the Structural Classification of Proteins (SCOP) [15]. We use the same 54 target families and the same test and training set splits as in the remote homology experiments in [7]. The sequences are 7329 SCOP domains obtained from version 1.59 of the database after purging with astral.stanford.edu so that no pair of sequences share more than 95% identity. Compared to [7], we reduce the number of available labeled training patterns by roughly a third. Data set sequences that were neither in the training nor test sets for experiments from [7] are included as unlabeled data. All methods are evaluated using the receiver operating characteristic (ROC) score and the ROC-50, which is the ROC score computed only up to the first 50 false positives. More details concerning the experimental setup can be found at `http://www1.cs.columbia.edu/compbio/svm-pairwise`.

In all experiments, we use an SVM classifier with a small soft margin parameter, set as in [7] . The SVM computations are performed using the freely available Spider Matlab machine learning package available at `http://www.kyb.tuebingen.mpg.de/bs/people/spider`. More information concerning the experiments, including data and source code scripts, can be found at `http://www.kyb.tuebingen.mpg.de/bs/people/weston/semiprot`.

**Semi-supervised setting.** Our first experiment shows that the neighborhood mismatch kernel makes better use of unlabeled data than the baseline method of "pulling in homologs" prior to training the SVM classifier, that is, simply finding close homologs of

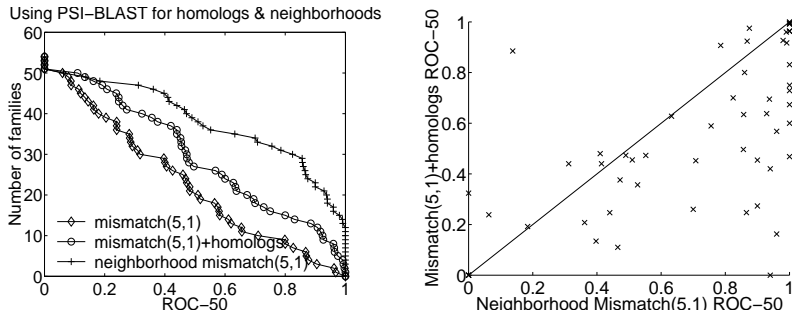

Figure 1: **Comparison of protein representations and classifiers using unlabeled data.**
The mismatch kernel is used to represent proteins, with close homologs being pulled in
from the unlabeled set with PSI-BLAST. Building a neighborhood with the neighborhood
mismatch kernel improves over the baseline of pulling in homologs.

|  | BLAST | | PSI-BLAST | |
|---|---|---|---|---|
|  | ROC-50 | ROC | ROC-50 | ROC |
| mismatch kernel | 0.416 | 0.870 | 0.416 | 0.870 |
| mismatch kernel + homologs | 0.480 | 0.900 | 0.550 | 0.910 |
| neighborhood mismatch kernel | 0.639 | 0.922 | 0.699 | 0.923 |

Table 1: Mean ROC-50 and ROC scores over 54 target families for semi-supervised exper-
iments, using BLAST and PSI-BLAST.

the positive training examples in the unlabeled set and adding them to the positive training
set for the SVM. Homologs come from the unlabeled set (not the test set), and "neigh-
bors" for the neighborhood kernel come from the training plus unlabeled data. We com-
pare the methods using the mismatch kernel representation with $k = 5$ and $m = 1$, as
used in [6]. Homologs are chosen via PSI-BLAST as having a pairwise score (E-value)
with any of the positive training samples less than 0.05, the default parameter setting [1].
The neighborhood mismatch kernel uses the same threshold to choose neighborhoods.
For the neighborhood kernel, we normalize before and after the averaging operation via
$K_{ij} \leftarrow K_{ij}/\sqrt{K_{ii}K_{jj}}$. The results are given in Figure 1 and Table 1. The former plots
the number of families achieving a given ROC-50 score, and a strongly performing method
thus produces a curve close to the top right of the plot. A signed rank test shows that the
neighborhood mismatch kernel yields significant improvement over adding homologs ($p$-
value 3.9e-05). Note that the PSI-BLAST scores in these experiments are built using the
whole database of 7329 sequences (that is, test sequences in a given experiment are also
available to the PSI-BLAST algorithm), so these results are slightly optimistic. However,
the comparison of methods in a truly inductive setting using BLAST shows the same im-
provement of the neighborhood mismatch kernel over adding homologs ($p$-value 8.4e-05).
Adding homologs to the (much larger) negative training set in addition to pulling in the pos-
itive homologs gives poorer performance than only adding the positive homologs (results
not shown).

**Transductive setting.**   In the following experiments, we consider a *transductive* setting,
in which the test points are given to the methods in advance as unlabeled data, giving
slightly improved results over the last section. Although this setting is unrealistic for a
real protein classification system, it more easily enables comparison with random walk
and spectral clustering kernels, which do not easily work in another setting. In Figure 2
(left), we again show the mismatch kernel compared with pulling in homologs and the
neighborhood kernel. This time we also compare with the bagged mismatch kernel using

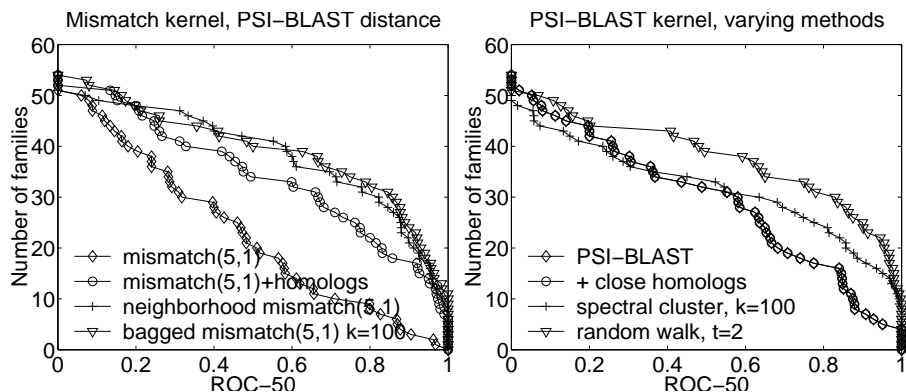

Figure 2: **Comparison of protein representations and classifiers using unlabeled data in a *transductive setting*.** Neighborhood and bagged mismatch kernels outperform pulling in close homologs (left) and equal or outperform previous semi-supervised methods (right).

| | ROC-50 | ROC | | ROC-50 | ROC |
|---|---|---|---|---|---|
| mismatch kernel | 0.416 | 0.875 | PSI-BLAST kernel | 0.533 | 0.866 |
| mismatch kernel + homologs | 0.625 | 0.924 | PSI-BLAST+homologs kernel | 0.585 | 0.873 |
| neighborhood mismatch kernel | 0.704 | 0.917 | spectral clustering kernel | 0.581 | 0.861 |
| bagged mismatch kernel ($k = 100$) | 0.719 | 0.943 | random walk kernel | 0.691 | 0.915 |
| bagged mismatch kernel ($k = 400$) | 0.671 | 0.935 | transductive SVM | 0.637 | 0.874 |

Table 2: Mean ROC-50 and ROC scores over 54 target families for *transductive* experiments.

bagged $k$-means with $k = 100$ and $n = 100$ runs, which gave the best results. We found the method quite insensitive to $k$. The result for $k = 400$ is also given in Table 2.

We then compare these methods to using random walk and spectral clustering kernels. Both methods do not work well for the mismatch kernel (see online supplement), perhaps because the feature vectors are so orthogonal. However, for a PSI-BLAST representation via empirical kernel map, the random walk outperforms pulling in homologs. We take the empirical map with $\Phi(x) = \langle \exp(-\lambda d(x_1, x)), \ldots, \exp(-\lambda(d(x_m, x)) \rangle$, where $d(x, y)$ are PSI-BLAST E-values and $\lambda = \frac{1}{1000}$, which improves over a linear map. We report results for the best parameter choices, $t = 2$ for the random walk and $k = 200$ for spectral clustering. We found the latter quite brittle with respect to the parameter choice; results for other parameters can be found on the supplemental web site. For pulling in close homologs, we take the empirical kernel map only for points in the training set and the chosen close homologs. Finally, we also run transductive SVMs. The results are given in Table 2 and Figure 2 (right). A signed rank test (with adjusted $p$-value cut-off of 0.05) finds no significant difference between the neighborhood kernel, the bagged kernel ($k = 100$), and the random walk kernel in this transductive setting. Thus the new techniques are comparable with random walk, but are feasible to calculate on full scale problems.

## 7   Discussion

Two of the most important issues in protein classication are representation of sequences and handling unlabeled data. Two developments in recent kernel methods research, string kernels and cluster kernels, address these issues separately. We have described two kernels — the *neighborhood mismatch kernel* and the *bagged mismatch kernel* — that combine

both approaches and yield state-of-the-art performance in protein classification. Practical use of semi-supervised protein classification techniques requires computational efficiency. Many cluster kernels require diagonalization of the full labeled plus unlabeled data kernel matrix. The neighborhood and bagged kernel approaches, used with an efficient string kernel, are fast and scalable cluster kernels for sequence data. Moreover, these techniques can be applied to any problem with a meaningful local similarity measure or distance function.

Future work will deal with additional challenges of protein classification: addressing the full multi-class problem, which potentially involves thousands of classes; handling very small classes with few homologs; and dealing with missing classes, for which no labeled examples exist.

### Acknowledgments

We would like to thank Eleazar Eskin for discussions that contributed to the neighborhood kernel and Olivier Chapelle and Navin Lal for their help with this work.

# References

[1] S. F. Altschul, W. Gish, W. Miller, E. W. Myers, and D. J. Lipman. A basic local alignment search tool. *Journal of Molecular Biology*, 215:403–410, 1990.

[2] T. Smith and M. Waterman. Identification of common molecular subsequences. *Journal of Molecular Biology*, 147:195–197, 1981.

[3] A. Krogh, M. Brown, I. Mian, K. Sjolander, and D. Haussler. Hidden markov models in computational biology: Applications to protein modeling. *Journal of Molecular Biology*, 235:1501–1531, 1994.

[4] J. Park, K. Karplus, C. Barrett, R. Hughey, D. Haussler, T. Hubbard, and C. Chothia. Sequence comparisons using multiple sequences detect twice as many remote homologues as pairwise methods. *Journal of Molecular Biology*, 284(4):1201–1210, 1998.

[5] T. Jaakkola, M. Diekhans, and D. Haussler. A discriminative framework for detecting remote protein homologies. *Journal of Computational Biology*, 2000.

[6] C. Leslie, E. Eskin, J. Weston, and W. S. Noble. Mismatch string kernels for SVM protein classification. *Neural Information Processing Systems 15*, 2002.

[7] C. Liao and W. S. Noble. Combining pairwise sequence similarity and support vector machines for remote protein homology detection. *Proceedings of RECOMB*, 2002.

[8] S. F. Altschul, T. L. Madden, A. A. Schaffer, J. Zhang, Z. Zhang, W. Miller, and D. J. Lipman. Gapped BLAST and PSI-BLAST: A new generation of protein database search programs. *Nucleic Acids Research*, 25:3389–3402, 1997.

[9] X. Zhu and Z. Ghahramani. Learning from labeled and unlabeled data with label propagation. Technical report, CMU, 2002.

[10] O. Chapelle, J. Weston, and B. Schoelkopf. Cluster kernels for semi-supervised learning. *Neural Information Processing Systems 15*, 2002.

[11] M. Szummer and T. Jaakkola. Partially labeled classification with Markov random walks. *Neural Information Processing Systems 14*, 2001.

[12] M. Seeger. Learning with labeled and unlabeled data. Technical report, University of Edinburgh, 2001.

[13] A. Ng, M. Jordan, and Y. Weiss. On spectral clustering: analysis and an algorithm. *Neural Processing Information Systems 14*, 2001.

[14] T. Joachims. Transductive inference for text classification using support vector machines. *Proceedings of ICML*, 1999.

[15] A. G. Murzin, S. E. Brenner, T. Hubbard, and C. Chothia. SCOP: A structural classification of proteins database for the investigation of sequences and structures. *Journal of Molecular Biology*, 247:536–540, 1995.
